# Learning Eigenvectors for Free

**Wouter M. Koolen**
Royal Holloway and CWI
wouter@cs.rhul.ac.uk

**Wojtek Kotłowski**
Centrum Wiskunde & Informatica
kotlowsk@cwi.nl

**Manfred K. Warmuth**
UC Santa Cruz
manfred@cse.ucsc.edu

## Abstract

We extend the classical problem of predicting a sequence of outcomes from a finite alphabet to the matrix domain. In this extension, the alphabet of $n$ outcomes is replaced by the set of all *dyads*, i.e. outer products $\boldsymbol{u}\boldsymbol{u}^\top$ where $\boldsymbol{u}$ is a vector in $\mathbb{R}^n$ of unit length. Whereas in the classical case the goal is to learn (i.e. sequentially predict as well as) the best multinomial distribution, in the matrix case we desire to learn the density matrix that best explains the observed sequence of dyads. We show how popular online algorithms for learning a multinomial distribution can be extended to learn density matrices. Intuitively, learning the $n^2$ parameters of a density matrix is much harder than learning the $n$ parameters of a multinomial distribution. Completely surprisingly, we prove that the worst-case regrets of certain classical algorithms and their matrix generalizations are identical. The reason is that the worst-case sequence of dyads share a common eigensystem, i.e. the worst case regret is achieved in the classical case. So these matrix algorithms learn the eigenvectors without any regret.

## 1   Introduction

We consider the extension of the classical online problem of predicting outcomes from a finite alphabet to the matrix domain. In this extension, the alphabet of $n$ outcomes is replaced by a set of all dyads, i.e. outer products $\boldsymbol{u}\boldsymbol{u}^\top$ where $\boldsymbol{u}$ is a unit vector in $\mathbb{R}^n$. Whereas classically the goal is to learn as well as the best multinomial distribution over outcomes, in the matrix case we desire to learn the distribution over dyads that best explains the sequence of dyads seen so far. A distribution on dyads is summarized as a *density matrix*, i.e. a symmetric positive-definite[1] matrix of unit trace. Such matrices are heavily used in quantum physics, where dyads represent states. We will show how popular online algorithms for learning multinomials can be extended to learn density matrices.

Considerable attention has been placed recently on generalizing algorithms for learning and optimization problems from probability vector parameters to density matrices [17, 19]. Efficient semidefinite programming algorithms have been devised [1] and better approximation algorithms for NP-hard problems have been obtained [2] by employing on-line algorithms that update a density matrix parameter. Also two important quantum complexity classes were shown to collapse based on these algorithms [8]. Even though the matrix generalization led to progress in many contexts, in the original domain of on-line learning, the regret bounds proven for the algorithms in the matrix case are often the same as those provable for the original classical finite alphabet case [17, 19]. Therefore it was posed as an open problem to determine whether this is just a case of loose classical bound or whether there truly exists a "free matrix lunch" for some of these algorithms [18]. Such algorithms essentially would learn the eigensystem of the data for free without incurring any additional regret. This is non-intuitive, since one would expect a matrix to have $n^2$ parameters and be much harder to learn than an $n$ dimensional parameter vector.

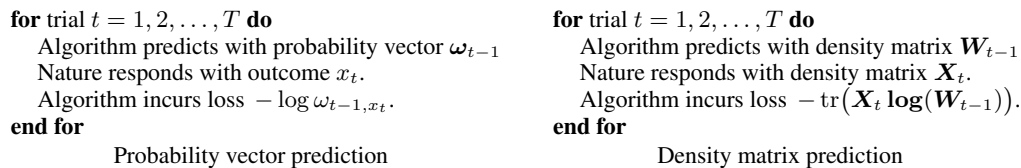

<div align="center">

| **for** trial $t = 1, 2, \ldots, T$ **do** | **for** trial $t = 1, 2, \ldots, T$ **do** |
|---|---|
| Algorithm predicts with probability vector $\boldsymbol{\omega}_{t-1}$ | Algorithm predicts with density matrix $\boldsymbol{W}_{t-1}$ |
| Nature responds with outcome $x_t$. | Nature responds with density matrix $\boldsymbol{X}_t$. |
| Algorithm incurs loss $-\log \omega_{t-1,x_t}$. | Algorithm incurs loss $-\operatorname{tr}\big(\boldsymbol{X}_t \log(\boldsymbol{W}_{t-1})\big)$. |
| **end for** | **end for** |
| Probability vector prediction | Density matrix prediction |

</div>

Figure 1: Protocols

In this paper we investigate this frivolously named but deep "free matrix lunch" question in arguably the simplest context: learning a multinomial distribution. In the classical case, there are $n \geq 2$ outcomes and a distribution is parametrized by an $n$-dimensional probability vector $\boldsymbol{\omega}$, where $\omega_i$ is the probability of outcome $i$. One can view the base vectors $\boldsymbol{e}_i$ as the elementary events and the probability vector as a mixture of these events: $\boldsymbol{\omega} = \sum_i \omega_i \boldsymbol{e}_i$. We define a "matrix generalization" of a multinomial which is parametrized by a density matrix $\boldsymbol{W}$ (positive matrix of unit trace). Now the elementary events are dyads of the form $\boldsymbol{u}\boldsymbol{u}^\top$, where $\boldsymbol{u}$ is a unit vector in $\mathbb{R}^n$. Dyads are the representations of states used in quantum physics [20]. A density matrix is a mixture of dyads. Whereas probability vectors represent uncertainty over $n$ basis vectors, density matrices can be viewed as representing uncertainty over infinitely many dyads in $\mathbb{R}^n$.

In the classical case, the algorithm predicts at trial $t$ with multinomial $\boldsymbol{\omega}_{t-1}$. Nature produces an outcome $x_t \in \{1, \ldots, n\}$, and the algorithm incurs loss $-\log(\omega_{t-1,x_t})$. The most common heuristic (a.k.a. the Laplace estimator) chooses $\omega_{t-1,i}$ proportional to 1 plus the number of previous trials in which outcome $i$ was observed. The on-line algorithms are evaluated by their worst-case regret over data sequences, where the regret is the additional loss of the algorithm over the total loss of the best probability vector chosen in hindsight.

In this paper we develop the corresponding matrix setting, where the algorithm predicts with a density matrix $\boldsymbol{W}_{t-1}$, Nature produces a dyad $\boldsymbol{x}_t \boldsymbol{x}_t^\top$, and the algorithm incurs loss $-\boldsymbol{x}_t^\top \log(\boldsymbol{W}_{t-1}) \boldsymbol{x}_t$. Here $\log$ denotes the matrix logarithm. We are particularly interested in how the regret changes when the algorithms are generalized to the matrix case. Surprisingly we can show that for the Laplace as well as the Krichevsky-Trofimov [10] estimators the worst-case regret is the same in the matrix case as it is in the classical case. For the Last-Step Minimax algorithm [16], we can prove the same regret bound for the matrix case that was proven for the classical case.

Why are we doing this? Most machine learning algorithms deal with vector parameters. The goal of this line of research is to develop methods for handling matrix parameters. We are used to dealing with probability vectors. Recently a probability calculus was developed for density matrices [20] including various Bayes rules for updating generalized conditionals. The vector problems are typically retained as special cases of the matrix problems, where the eigensystem is fixed and only the vectors of eigenvalues has to be learned. We exhibit for the first time a basic fundamental problem, for which the regret achievable in the matrix case is no higher than the regret achievable in the original vector setting.

**Paper outline** Definitions and notation are given in the next section, followed by proofs of the free matrix lunch for the three discussed algorithms in Section 3. At the core of our proofs is a new technical lemma for mixing quantum entropies. We also discuss the minimax algorithm for multinomials due to Shtarkov, and corresponding minimax algorithm for density matrices. We provide strong experimental evidence that the free matrix lunch holds for this algorithm as well. To put the results into context, we motivate and discuss our choice of the loss function, and compare it to several alternatives in Section 4. More discussion and perspective is provided in the Section 5.

## 2 Setup

The protocol for the classical *probability vector prediction* problem and the new *density matrix prediction* problem are displayed side-by-side in Figure 1. We explain the latter problem. Learning proceeds in trials. During trial $t$ the algorithm predicts with a density matrix $\boldsymbol{W}_{t-1}$. We use index $t-1$ to indicate that is based on the $t-1$ previous outcomes. Then nature responds with an outcome

density matrix $\boldsymbol{X}_t$. The discrepancy between prediction and outcome is measured by the *matrix entropic loss*

$$\ell(\boldsymbol{W}_{t-1}, \boldsymbol{X}_t) := -\operatorname{tr}\big(\boldsymbol{X}_t \log(\boldsymbol{W}_{t-1})\big), \tag{1}$$

where $\log$ denotes matrix logarithm[2]. When the outcome density matrix $\boldsymbol{X}_t$ is a dyad $\boldsymbol{x}_t \boldsymbol{x}_t^\top$, then this loss becomes $-\boldsymbol{x}_t^\top \log(\boldsymbol{W}_{t-1}) \boldsymbol{x}_t$, which is the simplified form of the entropic loss discussed in the introduction. Also if the prediction density matrix is diagonal, i.e. it has the form $\boldsymbol{W}_{t-1} = \sum_i \omega_{t-1,i} \, \boldsymbol{e}_i \boldsymbol{e}_i^\top$ for some probability vector $\boldsymbol{\omega}_{t-1}$, and the outcome $\boldsymbol{X}_t$ is an eigendyad $\boldsymbol{e}_j \boldsymbol{e}_j^\top$ of the same eigensystem, then this loss simplifies to the classical *log loss*: $\ell(\boldsymbol{W}_{t-1}, \boldsymbol{X}_t) = -\log(\omega_{t-1,j})$. The above definition is not the only way to promote the log loss to the matrix domain. Yet, in Section 4 we justify this choice.

We aim to design algorithms with low regret compared to the best fixed density matrix in hindsight. The loss of the best fixed density matrix can be expressed succinctly in terms of the *von Neumann entropy*, which is defined for any density matrix $\boldsymbol{D}$ as $H(\boldsymbol{D}) := -\operatorname{tr}(\boldsymbol{D} \log \boldsymbol{D})$, and the *sufficient statistic* $\boldsymbol{S}_T = \sum_{t=1}^T \boldsymbol{X}_t$ as follows: $\inf_{\boldsymbol{W}} \sum_{t=1}^T \ell(\boldsymbol{W}, \boldsymbol{X}_t) = T H\left(\frac{\boldsymbol{S}_T}{T}\right)$. For fixed data $\boldsymbol{X}_1, \ldots, \boldsymbol{X}_T$, the *regret* of a strategy that issues prediction $\boldsymbol{W}_t$ after observing $\boldsymbol{X}_1, \ldots, \boldsymbol{X}_t$ is

$$\sum_{t=1}^T \ell(\boldsymbol{W}_{t-1}, \boldsymbol{X}_t) - T H\left(\frac{\boldsymbol{S}_T}{T}\right), \tag{2}$$

and the *worst-case regret* on $T$ trials is obtained by taking $\sup_{\boldsymbol{X}_1, \ldots, \boldsymbol{X}_T}$ over (2). Our aim is to design strategies for density matrix prediction that have low worst-case regret.

## 3 Free Matrix Lunches

In this section, we will show how four popular online algorithms for learning multinomials can be extended to learning density matrices. We start with the simple Laplace estimator, continue with its improved version known as the Krichevsky-Trofimov estimator, and also extend the less known Last Step Minimax strategy which has even less regret. We will prove a version of the free matrix lunch (FML) for all three algorithms. Finally we discuss the minimax algorithm for which we have experimental evidence that the free matrix lunch holds as well.

### 3.1 Laplace

After observing classical data with sufficient statistic vector $\boldsymbol{\sigma}_t = \sum_{q=1}^t \boldsymbol{e}_{x_q}$, *classical Laplace* predicts with the probability vector $\boldsymbol{\omega}_t := \frac{\boldsymbol{\sigma}_t + \mathbf{1}}{t+n}$ consisting of the normalized smoothed counts. By analogy, after observing matrix data with sufficient statistic $\boldsymbol{S}_t = \sum_{q=1}^t \boldsymbol{X}_t$, *matrix Laplace* predicts with the correspondingly smoothed matrix $\boldsymbol{W}_t := \frac{\boldsymbol{S}_t + \boldsymbol{I}}{t+n}$. Classical Laplace is commonly motivated as either the Bayes predictive distribution w.r.t. the uniform prior or as a loss minimization with virtual outcomes [3]. The latter motivation can be "lifted" to the matrix domain by adding $n$ virtual outcomes at $\boldsymbol{I}/n$:

$$\boldsymbol{W}_t = \operatorname*{argmin}_{\boldsymbol{W} \text{ dens. mat.}} \left\{ n \, \ell(\boldsymbol{W}, \boldsymbol{I}/n) + \sum_{q=1}^t \ell(\boldsymbol{W}, \boldsymbol{X}_q) \right\} = \frac{\boldsymbol{S}_t + \boldsymbol{I}}{t+n}. \tag{3}$$

The worst-case regret of classical Laplace after $T$ iterations equals $\log \binom{T+n-1}{n-1} \le (n-1)\log(T+1)$ (see e.g. [6]). We now show that in the matrix case, no additional regret is incurred.

**Theorem 1** (Laplace FML). *The worst-case regrets of classical and matrix Laplace coincide.*

*Proof.* Let $\boldsymbol{W}_t^*$ denote the best density matrix for the first $t$ outcomes. The regret (2) of matrix Laplace can be bounded as follows:

$$\sum_{t=1}^T \ell(\boldsymbol{W}_{t-1}, \boldsymbol{X}_t) - \sum_{t=1}^T \ell(\boldsymbol{W}_T^*, \boldsymbol{X}_t) \le \sum_{t=1}^T \Big( \ell(\boldsymbol{W}_{t-1}, \boldsymbol{X}_t) - \ell(\boldsymbol{W}_t^*, \boldsymbol{X}_t) \Big). \tag{4}$$

Now consider each term in the right-hand sum separately. The $t^{\text{th}}$ term equals

$$-\text{tr}\left(\boldsymbol{X}_t\left(\log\frac{\boldsymbol{S}_{t-1}+\boldsymbol{I}}{t-1+n}-\log\frac{\boldsymbol{S}_t}{t}\right)\right) \;=\; \log\left(\frac{t-1+n}{t}\right)-\text{tr}\left(\boldsymbol{X}_t(\log(\boldsymbol{S}_{t-1}+\boldsymbol{I})-\log\boldsymbol{S}_t)\right).$$

Note that the first term constitutes the "classical" part of the per-round regret, while the second term is the "matrix" part. The matrix part is non-positive since $\boldsymbol{S}_{t-1}+\boldsymbol{I}\succeq\boldsymbol{S}_t$, and the logarithm is a matrix monotone operation (i.e. $\boldsymbol{A}\succeq\boldsymbol{B}$ implies $\log\boldsymbol{A}\succeq\log\boldsymbol{B}$). By omitting it, we obtain an upper bound on the regret of matrix Laplace, that is tight: for any sequence of identical dyads the matrix part is zero and (4) holds with equality since $\boldsymbol{W}_t^*=\boldsymbol{W}_T^*$ for all $t\leq T$. The same upper bound is also met by classical Laplace on any sequence of identical outcomes [6]. $\qquad\square$

We just showed that matrix Laplace has the same worst-case regret as classical Laplace, albeit matrix Laplace learns a matrix of $n^2$ parameters whereas classical Laplace only learns $n$ probabilities. No additional regret is incurred for learning the eigenvectors. Matrix Laplace can update $\boldsymbol{W}_t$ in $O(n^2)$ time per trial. The same will be true for our next algorithm.

### 3.2 Krichevsky-Trofimov (KT)

*Classical* and *matrix KT* smooth by adding $\frac{1}{2}$ to each count, i.e. $\boldsymbol{\omega}_t:=\frac{\boldsymbol{\sigma}_t+1/2}{t+n/2}$ and $\boldsymbol{W}_t:=\frac{\boldsymbol{S}_t+\boldsymbol{I}/2}{t+n/2}$. The former can again be obtained as the Bayes predictive distribution w.r.t. Jeffreys' prior, the latter as the solution to the matrix entropic loss minimization problem (3) with $n/2$ virtual outcomes instead of $n$ for Laplace.

The leading term in the worst-case regret for classical KT is the *optimal* $\frac{1}{2}\log(T)$ rate per parameter instead of the $\log(T)$ rate for Laplace. More precisely, classical KT's worst-case regret after $T$ iterations is known to be $\log\frac{\Gamma(T+n/2)}{\Gamma(T+1/2)}+\log\frac{\Gamma(1/2)}{\Gamma(n/2)}\leq\frac{n-1}{2}\big(\log(T+1)+\log(\pi)\big)$ (see e.g. [6]). Again we show that no additional regret is incurred in the matrix case.

**Theorem 2** (KT FML). *The worst-case regrets of classical and matrix KT coincide.*

The proof uses the following key entropy decomposition lemma (proven in Appendix A):

**Lemma 1.** *For positive matrices $\boldsymbol{A},\boldsymbol{B}$ with $\boldsymbol{A}=\sum_i\alpha_i\,\boldsymbol{a}_i\boldsymbol{a}_i^\top$ the eigendecomposition of $\boldsymbol{A}$:*

$$H(\boldsymbol{A}+\boldsymbol{B})\;\geq\;\sum_{i=1}^n\frac{\boldsymbol{a}_i^\top\boldsymbol{B}\boldsymbol{a}_i}{\text{tr}(\boldsymbol{B})}\,H\big(\boldsymbol{A}+\text{tr}(\boldsymbol{B})\,\boldsymbol{a}_i\boldsymbol{a}_i^\top\big),$$

*Proof of Theorem 2.* We start by telescoping the regret (2) of matrix KT as follows

$$\sum_{t=1}^T\left(-\text{tr}\big(\boldsymbol{X}_t\log(\boldsymbol{W}_{t-1})\big)-tH\left(\frac{\boldsymbol{S}_{t-1}+\boldsymbol{X}_t}{t}\right)+(t-1)H\left(\frac{\boldsymbol{S}_{t-1}}{t-1}\right)\right). \tag{5}$$

We bound each term separately. Let us denote the eigendecomposition of $\boldsymbol{S}_{t-1}$ by $\boldsymbol{S}_{t-1}=\sum_{i=1}^n\sigma_i\,\boldsymbol{s}_i\boldsymbol{s}_i^\top$. Notice that since $\boldsymbol{W}_{t-1}$ plays in the eigensystem of $\boldsymbol{S}_{t-1}$, we have:

$$-\text{tr}\big(\boldsymbol{X}_t\log(\boldsymbol{W}_{t-1})\big)\;=\;-\text{tr}\Big(\boldsymbol{X}_t\sum_{i=1}^n\log(\omega_{t-1,i})\,\boldsymbol{s}_i\boldsymbol{s}_i^\top\Big)\;=\;-\sum_{i=1}^n\boldsymbol{s}_i^\top\boldsymbol{X}_t\boldsymbol{s}_i\log(\omega_{t-1,i}).$$

Moreover, it follows from Lemma 1 that:

$$H\left(\frac{\boldsymbol{S}_{t-1}+\boldsymbol{X}_t}{t}\right)\;\geq\;\sum_{i=1}^n\boldsymbol{s}_i^\top\boldsymbol{X}_t\boldsymbol{s}_iH\left(\frac{\boldsymbol{S}_{t-1}+\boldsymbol{s}_i\boldsymbol{s}_i^\top}{t}\right).$$

Taking this equality and inequality into account, the $t^{\text{th}}$ term in (5) is bounded above by:

$$\delta_t\;:=\;\sum_{i=1}^n\boldsymbol{s}_i^\top\boldsymbol{X}_t\boldsymbol{s}_i\left(-\log(\omega_{t-1,i})-tH\left(\frac{\boldsymbol{S}_{t-1}+\boldsymbol{s}_i\boldsymbol{s}_i^\top}{t}\right)+(t-1)H\left(\frac{\boldsymbol{S}_{t-1}}{t-1}\right)\right), \tag{6}$$

which, in turn, is at most:

$$\delta_t\;\leq\;\sup_i\left(-\log(\omega_{t-1,i})-tH\left(\frac{\boldsymbol{S}_{t-1}+\boldsymbol{s}_i\boldsymbol{s}_i^\top}{t}\right)+(t-1)H\left(\frac{\boldsymbol{S}_{t-1}}{t-1}\right)\right).$$

In other words the per-round regret increase is largest for one of the eigenvectors of the sufficient statistic $\boldsymbol{S}_{t-1}$, i.e. for classical data. To get an upper bound, maximize over $\boldsymbol{S}_0, \ldots, \boldsymbol{S}_{T-1}$ independently, each with the constraint that $\operatorname{tr}(\boldsymbol{S}_t) = t$. A particular maximizer is $\boldsymbol{S}_t = t\, \boldsymbol{e}_1 \boldsymbol{e}_1^\top$, which is the sufficient statistic of the sequence of outcomes all equal to $\boldsymbol{X}_t = \boldsymbol{e}_1 \boldsymbol{e}_1^\top$. For this sequence all bounding steps hold with equality. Hence the matrix KT regret is below the classical KT regret. The reverse is obvious. $\qquad\square$

### 3.3 Last Step Minimax

The bounding technique, developed using Lemma 1 and applied to KT can be used to prove bounds for a much broader class of prediction strategies. The crucial part of the KT proof was showing that each term in the telescoped regret (5) can be bounded above by $\delta_t$ as defined in (6), in which all matrices share the same eigensystem, and which is hence equivalent to the corresponding classical expression. The only property of the prediction strategy that we used was that it plays in the eigensystem of the past sufficient statistic. Therefore, using the same line of argument, we can show that if for some classical prediction strategy we can obtain a meaningful regret bound by bounding each term in the regret $\delta_t$ independently, we can obtain *the same* bound for the corresponding matrix strategy, i.e. its spectral promotion.

In particular, we can push this argument to its limit by considering the algorithm *designed* to minimize $\delta_t$ in each iteration. This algorithm is known as Last Step Minimax.

In fact, the *Last Step Minimax (LSM) principle* is a general recipe for online prediction, which states that the algorithm should minimize the worst-case regret with respect to the next outcome [16]. In other words, it should act as the minimax algorithm given that the time horizon is one iteration ahead. In the classical case for the multinomial distribution, after observing data with sufficient statistic $\boldsymbol{\sigma}_{t-1}$, *classical LSM* predicts with

$$\boldsymbol{\omega}_{t-1} \;:=\; \operatorname*{argmin}_{\boldsymbol{\omega}} \max_{x_t} \left\{ \underbrace{\ell(\boldsymbol{\omega}, x_t)}_{-\log(\omega_{t-1,x_t})} - \underbrace{\sum_{q=1}^{t} \ell(\boldsymbol{\omega}_t^*, x_q)}_{tH\left(\frac{\boldsymbol{\sigma}_t}{t}\right)} \right\} \;=\; \sum_{i=1}^{n} \frac{\exp\!\left(-tH\!\left(\frac{\boldsymbol{\sigma}_{t-1}+\boldsymbol{e}_i}{t}\right)\right)}{\sum_j \exp\!\left(-tH\!\left(\frac{\boldsymbol{\sigma}_{t-1}+\boldsymbol{e}_j}{t}\right)\right)} \boldsymbol{e}_i. \quad (7)$$

Classical LSM is analyzed in [16] for the Bernoulli ($n = 2$) case. For our straightforward generalization to the classical multinomial case, the regret is bounded by $\frac{n-1}{2}\ln(T+1) + 1$. LSM is therefore slightly better than KT.

Applying the Last Step Minimax principle to density prediction, we obtain *matrix LSM* which issues prediction:

$$\boldsymbol{W}_{t-1} \;:=\; \operatorname*{argmin}_{\boldsymbol{W}} \max_{\boldsymbol{X}_t} \left\{ -\operatorname{tr}\!\big(\boldsymbol{X}_t \log(\boldsymbol{W})\big) - tH\!\left(\frac{\boldsymbol{S}_t}{t}\right) \right\}.$$

We show that matrix LSM learns the eigenvectors without additional regret.

**Theorem 3** (LSM FML). *The regrets of classical and matrix LSM are at most $\frac{n-1}{2}\ln(T+1) + 1$.*

*Proof.* We determine the form of $\boldsymbol{W}_{t-1}$. By Sion's minimax theorem [15]:

$$\min_{\boldsymbol{W}} \max_{\boldsymbol{X}_t} \left\{ -\operatorname{tr}\!\big(\boldsymbol{X}_t \log(\boldsymbol{W})\big) - tH\!\left(\frac{\boldsymbol{S}_t}{t}\right) \right\} \;=\; \max_{P} \min_{\boldsymbol{W}} \mathbb{E}_P\!\left[ -\operatorname{tr}\!\big(\boldsymbol{X}_t \log(\boldsymbol{W})\big) - tH\!\left(\frac{\boldsymbol{S}_t}{t}\right) \right],$$

where $P$ ranges over probability distribution on density matrices $\boldsymbol{X}_t$. Plugging in the minimizer $\boldsymbol{W} = \mathbb{E}_P[\boldsymbol{X}_t]$, the right hand side becomes:

$$\max_{P} \left\{ H\!\big(\mathbb{E}_P[\boldsymbol{X}_t]\big) - \mathbb{E}_P\!\left[ tH\!\left(\frac{\boldsymbol{S}_t}{t}\right) \right] \right\}. \quad (8)$$

Now decompose $\boldsymbol{S}_{t-1}$ as $\sum_{i=1}^{n} \sigma_i\, \boldsymbol{s}_i \boldsymbol{s}_i^\top$. Using Lemma 1, we can bound the second expression inside the maximum:

$$\mathbb{E}_P\!\left[ tH\!\left(\frac{\boldsymbol{S}_t}{t}\right) \right] \geq \mathbb{E}_P\!\left[ t\sum_{i=1}^{n} \boldsymbol{s}_i^\top \boldsymbol{X}_t \boldsymbol{s}_i H\!\left(\frac{\boldsymbol{S}_{t-1} + \boldsymbol{s}_i \boldsymbol{s}_i^\top}{t}\right) \right] \;=\; t\sum_{i=1}^{n} \boldsymbol{s}_i^\top \mathbb{E}_P[\boldsymbol{X}_t]\, \boldsymbol{s}_i H\!\left(\frac{\boldsymbol{S}_{t-1} + \boldsymbol{s}_i \boldsymbol{s}_i^\top}{t}\right).$$

On the other hand, we know that the entropy does not decrease when we replace the argument $\mathbb{E}_P[\boldsymbol{X}_t]$ by its pinching (a.k.a. projective measurement) $\sum_{i=1}^n (\boldsymbol{u}_i^\top \mathbb{E}_P[\boldsymbol{X}_t]\boldsymbol{u}_i)\,\boldsymbol{u}_i\boldsymbol{u}_i^\top$ w.r.t. any eigensystem $\boldsymbol{u}_i$ [12]. Therefore, we have:

$$H\big(\mathbb{E}_P[\boldsymbol{X}_t]\big) \;\leq\; H\left(\sum_{i=1}^n (\boldsymbol{s}_i^\top \mathbb{E}_P[\boldsymbol{X}_t]\boldsymbol{s}_i)\,\boldsymbol{s}_i\boldsymbol{s}_i^\top\right) \;=\; H(\boldsymbol{p}),$$

where the last entropy is a classical entropy and $\boldsymbol{p}$ is a vector such that $p_i = \boldsymbol{s}_i^\top \mathbb{E}_P[\boldsymbol{X}_t]\boldsymbol{s}_i$. Combining those two results together, we have:

$$H\big(\mathbb{E}_P[\boldsymbol{X}_t]\big) - \mathbb{E}_P\left[tH\left(\frac{\boldsymbol{S}_t}{t}\right)\right] \;\leq\; H(\boldsymbol{p}) - t\sum_{i=1}^n p_i H\left(\frac{\boldsymbol{\sigma}_{t-1}+\boldsymbol{e}_i}{t}\right).$$

Note that we have equality only when the distribution $P$ puts nonzero mass only on the eigenvectors $\boldsymbol{s}_1,\ldots,\boldsymbol{s}_n$. This means that when $\boldsymbol{p}$ is fixed, we will maximize (8) by using a distribution with such a property, i.e. $P$ is restricted to the eigensystem of $\boldsymbol{S}_{t-1}$. This, in turn, means that $\boldsymbol{W}_{t-1} = \mathbb{E}_P[\boldsymbol{X}_t]$ will play in the eigensystem of $\boldsymbol{S}_{t-1}$ as well. It follows that $\boldsymbol{W}_{t-1}$ is the classical LSM strategy in the eigensystem of $\boldsymbol{S}_{t-1}$, i.e. $\boldsymbol{W}_{t-1} = \sum_i \omega_{t-1,i}\,\boldsymbol{s}_i\boldsymbol{s}_i^\top$, where $\boldsymbol{\omega}_{t-1}$ are taken as in (7).

The proof of the classical LSM guarantee is based on bounding the per-round regret increase:

$$\delta_t \;:=\; -\log(\omega_{t-1,x_t}) - tH\left(\frac{\boldsymbol{\sigma}_{t-1}+\boldsymbol{e}_{x_t}}{t}\right) + (t-1)H\left(\frac{\boldsymbol{\sigma}_{t-1}}{t-1}\right),$$

by choosing the worst case w.r.t. $x_t$ and $\boldsymbol{\sigma}_{t-1}$. Since, for matrices, the worst case for the corresponding matrix version of $\delta_t$, see (6), is the diagonal case, the whole analysis immediately goes through and we get the same bound as for classical LSM. $\qquad\square$

Note that the bound for LSM is *not* tight, i.e. there exists no data sequence for which the bound is achieved. Therefore, the bound for matrix LSM is also not tight. This theorem is a weaker FML because it only relates worst-case regret *bounds*. We have verified experimentally that the actual regrets coincide in dimension $n = 2$ for up to $T = 5$ outcomes, using a grid of 30 dyads per trial, with uniformly spaced $(\boldsymbol{x}^\top \boldsymbol{e}_1)^2$. So we believe that in fact

**Conjecture 1** (LSM FML). *The worst-case regrets of classical and matrix LSM coincide.*

To execute the LSM matrix strategy, we need to have the eigendecomposition of the sufficient statistic. For density matrix data $\boldsymbol{X}_t$, we may need to recompute it each trial in $\Omega(n^3)$ time. For dyadic data $\boldsymbol{x}_t\boldsymbol{x}_t^\top$ it can be incrementally updated in $O(n^2)$ per trial with methods along the lines of [11].

### 3.4 Shtarkov

Fix horizon $T$. The minimax algorithm for multinomials, due to Shtarkov [14], minimizes the worst-case regret

$$\inf_{\boldsymbol{\omega}_0}\sup_{x_1}\ldots\inf_{\boldsymbol{\omega}_{T-1}}\sup_{x_T}\sum_{t=1}^T \ell(\boldsymbol{\omega}_{t-1},x_t) - TH\left(\frac{\boldsymbol{\sigma}_T}{T}\right). \tag{9}$$

After observing data with sufficient statistic $\boldsymbol{\sigma}_t$ and hence with $r := T - t$ rounds remaining, *classical Shtarkov* predicts with

$$\boldsymbol{\omega}_t \;:=\; \sum_{i=1}^n \frac{\phi_{r-1}(\boldsymbol{\sigma}_t+\boldsymbol{e}_i)}{\phi_r(\boldsymbol{\sigma}_t)}\boldsymbol{e}_i \quad\text{where}\quad \phi_r(\boldsymbol{\sigma}) \;:=\; \sum_{\substack{c_1,\ldots,c_n\\ \sum_{i=1}^n c_i=r}} \binom{r}{c_1,\ldots,c_n}\exp\left(-TH\left(\frac{\boldsymbol{\sigma}+\boldsymbol{c}}{T}\right)\right). \tag{10}$$

The so-called Shtarkov sum $\phi_r$ can be evaluated in time $O\big(n\,r\log(r)\big)$ using a straightforward extension of the method described in [9] for computing $\phi_T(\boldsymbol{0})$, which is based on dynamic programming and Fast Fourier Transforms.

The regret of classical Shtarkov equals $\log\phi_T(\boldsymbol{0}) \approx \frac{n-1}{2}\big(\log(T) - \log(n-2) + 1\big)$ [6]. This is again better than Last Step Minimax, which is in turn better than KT which dominates Laplace.

The minimax algorithm for density matrices, called *matrix Shtarkov*, optimizes the worst-case regret

$$\inf_{\boldsymbol{W}_0} \sup_{\boldsymbol{X}_1} \dots \inf_{\boldsymbol{W}_{T-1}} \sup_{\boldsymbol{X}_T} \sum_{t=1}^{T} \ell(\boldsymbol{W}_{t-1}, \boldsymbol{X}_t) - TH\left(\frac{\boldsymbol{S}_T}{T}\right). \tag{11}$$

To this end, after observing data with sufficient statistic $\boldsymbol{S}_t$, with $r$ rounds remaining, it predicts with

$$\boldsymbol{W}_t := \operatorname*{argmin}_{\boldsymbol{W}} \sup_{\boldsymbol{X}} \ell(\boldsymbol{W}, \boldsymbol{X}) + R_{r-1}(\boldsymbol{S}_t + \boldsymbol{X}),$$

where $R_r$ is the tail sequence of inf/sups of (11) of length $r$. We now argue that the FML holds for matrix Shtarkov. Matrix Shtarkov is surprisingly difficult to analyze. However, we provide a simplifying conjecture that we verified experimentally. A rigorous proof remains an open problem. Our conjecture is that Lemma 1 holds with the entropy $H$ replaced by the minimax regret tail $R_r$:

**Conjecture 2.** *For each integer $r$, for each pair of positive matrices $\boldsymbol{A}$ and $\boldsymbol{B}$*

$$R_r(\boldsymbol{A} + \boldsymbol{B}) \geq \sum_i \frac{\boldsymbol{a}_i^\top \boldsymbol{B} \boldsymbol{a}_i}{\operatorname{tr}(\boldsymbol{B})} R_r\big(\boldsymbol{A} + \operatorname{tr}(\boldsymbol{B})\, \boldsymbol{a}_i \boldsymbol{a}_i^\top\big).$$

Note that this conjecture generalizes Lemma 1, which is retained as the case $r = 0$. It follows from this conjecture, using the same argument as for LSM, that matrix Shtarkov predicts in the eigensystem of $\boldsymbol{S}_t$, i.e. with $\boldsymbol{W}_t = \sum_i \omega_{t,i}\, \boldsymbol{s}_i \boldsymbol{s}_i^\top$, where $\boldsymbol{\omega}_t$ as in (10), and furthermore that

**Conjecture 3** (Shtarkov FML). *The worst-case regrets of classical and matrix Shtarkov coincide.*

We have verified Conjecture 3 for the matrix Bernoulli case ($n = 2$) up to $T = 5$ outcomes, using a grid of 30 dyads per trial, with uniformly spaced $(\boldsymbol{x}^\top \boldsymbol{e}_1)^2$. Then assuming that $R_r(\boldsymbol{S}) = \log(\phi(\boldsymbol{\sigma}))$, where $\boldsymbol{\sigma}$ are the eigenvalues of $\boldsymbol{S}$, for each $n$ from 2 to 5 we drew $10^5$ trace pairs uniformly from $[0, 10]$, then drew matrix pairs $\boldsymbol{A}$ and $\boldsymbol{B}$ uniformly at random with those traces. Conjecture 2 always held.

Obtaining the FML for the minimax algorithm is mathematically challenging and of academic interest but of minor practical relevance. First, the time horizon $T$ must be specified in advance, so the minimax algorithm can not be used in a purely online fashion. Secondly, the running time is *superlinear* in the number of rounds remaining, while it is *constant* for the previous three algorithms.

## 4 Motivation and Discussion of the Loss Function

The matrix entropic loss (1) that we choose as our loss function has a coding interpretation and it is a proper scoring rule. The latter seems to be a necessary condition for the free matrix lunch.

**Quantum coding** Classical log-loss forecasting can be motivated from the point of view of data compression and variable-length coding [7]. In information theory, the Kraft-McMillan inequality states that, ignoring rounding issues, for every uniquely decodable code with a code length function $\boldsymbol{\lambda}$, there is a probability distribution $\boldsymbol{\omega}$ such that $\lambda_i = -\log \omega_i$ for all symbols $i = 1, \dots, n$, and vice versa. Therefore, the log loss can be interpreted as the code length assigned to the observed outcome. Quantum information theory[13, 5] generalizes variable length coding to the quantum/density matrix case. Instead of messages composed of bits, the sender and the receiver exchange messages described by density matrices, and the role analogous to the message length is now played by the dimension of the density matrix. Variable-length quantum coding requires the definition of a *code length operator* $\boldsymbol{L}$, which is a positive matrix such that for any density matrix $\boldsymbol{X}$, $\operatorname{tr}(\boldsymbol{X}\boldsymbol{L})$ gives the expected dimension ("length") of the message assigned to $\boldsymbol{X}$. The quantum version of Kraft's inequality states that, ignoring rounding issues, for every variable-length quantum code with code-length operator $\boldsymbol{L}$, there exists a density matrix $\boldsymbol{W}$ such that $\boldsymbol{L} = -\log \boldsymbol{W}$. Therefore, the matrix entropic loss can be interpreted as the (expected) code length of the observed outcome.

**Proper score function** In decision theory, the loss function $\ell(\boldsymbol{\omega}, x)$ assessing the quality of predictions is also referred to as a *score function*. A score function is said to be *proper*, if for any distribution $\boldsymbol{p}$ on outcomes, the expected loss is minimized by predicting with $\boldsymbol{p}$ itself, i.e. $\operatorname{argmin}_{\boldsymbol{\omega}} \mathbb{E}_{x \sim \boldsymbol{p}}[\ell(\boldsymbol{\omega}, x)] = \boldsymbol{p}$. Minimization of a proper score function leads to well-calibrated forecasting. The log loss is known to be a proper score function [4].

We will say that a matrix loss function $\ell(\boldsymbol{W}, \boldsymbol{X})$ is *proper* if for any distribution $P$ on density matrix outcomes, the expected loss with respect to $P$ is minimized by predicting with the mean outcome of $P$, i.e. $\operatorname{argmin}_{\boldsymbol{W}} \mathbb{E}_{\boldsymbol{X} \sim P}[\ell(\boldsymbol{W}, \boldsymbol{X})] = \mathbb{E}_{\boldsymbol{X} \sim P}[\boldsymbol{X}]$. The matrix entropic loss (1) is proper, for $\mathbb{E}_{\boldsymbol{X} \sim P}[-\operatorname{tr}(\boldsymbol{X} \log \boldsymbol{W})] = -\operatorname{tr}(\mathbb{E}_{\boldsymbol{X} \sim P}[\boldsymbol{X}] \log \boldsymbol{W})$ is minimized at $\boldsymbol{W} = \mathbb{E}_{\boldsymbol{X} \sim P}[\boldsymbol{X}]$ [12]. Therefore, minimization of the matrix entropic loss leads to well-calibrated forecasting, as in the classical case.

A second generalization of the log loss to the matrix domain used in quantum physics [12] is the *log trace loss* $\ell(\boldsymbol{W}, \boldsymbol{X}) := -\log(\operatorname{tr}(\boldsymbol{X} \boldsymbol{W}))$. Note that here the trace and the logarithm are exchanged compared to (1). The expression $\operatorname{tr}(\boldsymbol{X} \boldsymbol{W})$ plays an important role in quantum physics as the expected value of a measurement outcome, and for $\boldsymbol{X} = \boldsymbol{x} \boldsymbol{x}^\top$, $\operatorname{tr}(\boldsymbol{x} \boldsymbol{x}^\top \boldsymbol{W})$ is interpreted as a probability. However, log trace loss is not proper. The counterexample is straightforward: if we take $P$ uniform on $\{\boldsymbol{x}_1 \boldsymbol{x}_1^\top, \boldsymbol{x}_2 \boldsymbol{x}_2^\top\}$, then the minimizer of the expected log trace loss is $\boldsymbol{W} \propto (\boldsymbol{x}_1 + \boldsymbol{x}_2)(\boldsymbol{x}_1 + \boldsymbol{x}_2)^\top$, which differs from $\mathbb{E}_{\boldsymbol{X} \sim P}[\boldsymbol{X}] = \frac{1}{2}(\boldsymbol{x}_1 \boldsymbol{x}_1^\top + \boldsymbol{x}_2 \boldsymbol{x}_2^\top)$. Also for log trace loss we found an example (not presented) against the FML for the minimax algorithm.

A third generalization of the loss is $\ell(\boldsymbol{W}, \boldsymbol{X}) := -\log(\operatorname{tr}(\boldsymbol{X} \odot \boldsymbol{W}))$, where $\odot$ denotes the commutative "product" between matrices that underlies the probability calculus of [20].[3] This loss upper bounds the log trace loss. We don't know whether it is a proper scoring function. However, it equals the matrix entropic loss when $\boldsymbol{X}$ is a dyad.

Finally, another loss explored in the on-line learning community is the *trace loss* $\ell(\boldsymbol{W}, \boldsymbol{X}) := \operatorname{tr}(\boldsymbol{W} \boldsymbol{X})$. This loss is not a proper scoring function (it behaves like the absolute loss in the vector case) and we have an example that shows that there is no FML for the minimax algorithm in this case (not presented).

In summary, for there to exist a FML, properness of the loss function seems to be required.

## 5 Conclusion

We showed that the free matrix lunch holds for the matrix version of the KT estimator. Thus the conjectured free matrix lunch [18] is realized. Our paper raises many open questions. Perhaps the main one is whether the free matrix lunch holds for the matrix minimax algorithm. Also we would like to know what properties of the loss function and algorithm cause the free matrix lunch to occur. From the examples given in this paper it is tempting to believe that you always get a free matrix lunch when upgrading any classical sufficient-statistics-based predictor to a matrix version by just playing this predictor in the eigensystem of the current matrix sufficient statistics. However the following counter example shows that a general reduction must be more subtle: Consider *floored KT*, which predicts with $\omega_{t,i} \propto \lfloor \sigma_{t,i} \rfloor + 1/2$. For $T = 5$ trials in dimension $n = 2$, the worst-case regret is 1.297 for the classical log loss and 1.992 for matrix entropic loss.

## A  Proof of Lemma 1

We prove the following slightly stronger inequality for all $\gamma \geq 0$. The lemma is the case $\gamma = 1$.

$$f(\gamma) := H(\boldsymbol{A} + \gamma \boldsymbol{B}) - \sum_{i=1}^n \frac{\boldsymbol{a}_i^\top \boldsymbol{B} \boldsymbol{a}_i}{\operatorname{tr}(\boldsymbol{B})} \, H(\boldsymbol{A} + \gamma \operatorname{tr}(\boldsymbol{B}) \boldsymbol{a}_i \boldsymbol{a}_i^\top) \geq 0.$$

Since $f(0) = 0$, it suffices to show that $f'(\gamma) \geq 0$. Since $\frac{\partial H(\boldsymbol{D})}{\partial \boldsymbol{D}} = -\log(\boldsymbol{D}) - \boldsymbol{I}$,

$$f'(\gamma) = -\operatorname{tr}(\boldsymbol{B} \log(\boldsymbol{A} + \gamma \boldsymbol{B})) + \sum_{i=1}^n \boldsymbol{a}_i^\top \boldsymbol{B} \boldsymbol{a}_i \operatorname{tr}(\boldsymbol{a}_i \boldsymbol{a}_i^\top \log(\boldsymbol{A} + \gamma \operatorname{tr}(\boldsymbol{B}) \boldsymbol{a}_i \boldsymbol{a}_i^\top))$$

$$= \operatorname{tr}(\boldsymbol{B} \log(\boldsymbol{A} + \gamma \operatorname{tr}(\boldsymbol{B}) \boldsymbol{I})) - \operatorname{tr}(\boldsymbol{B} \log(\boldsymbol{A} + \gamma \boldsymbol{B})).$$

Since $\operatorname{tr}(\boldsymbol{B}) \boldsymbol{I} \succeq \boldsymbol{B}$, we have $\boldsymbol{A} + \gamma \operatorname{tr}(\boldsymbol{B}) \boldsymbol{I} \succeq \boldsymbol{A} + \gamma \boldsymbol{B}$, and hence the matrix monotonicity of the logarithm implies that $\log(\boldsymbol{A} + \gamma \operatorname{tr}(\boldsymbol{B}) \boldsymbol{I}) \succeq \log(\boldsymbol{A} + \gamma \boldsymbol{B})$, so that $f'(\gamma) \geq 0$. $\qquad \square$

## Footnotes

[1] We use positive in the non-strict sense, and omit 'symmetric' and 'definite'. Our matrices are real-valued.

[2]For any positive matrix with eigendecomposition $\boldsymbol{A} = \sum_i \alpha_i \, \boldsymbol{a}_i \boldsymbol{a}_i^\top$, $\log(\boldsymbol{A}) := \sum_i \log(\alpha_i) \, \boldsymbol{a}_i \boldsymbol{a}_i^\top$.

[3] We can compute $\boldsymbol{A} \odot \boldsymbol{B}$ as the matrix exponential of the sum of matrix logarithms of $\boldsymbol{A}$ and $\boldsymbol{B}$.

# References

[1] S. Arora, E. Hazan, and S. Kale. Fast algorithms for approximate semidefinite programming using the multiplicative weights update method. In *FOCS*, pages 339–348, 2005.

[2] S. Arora and S. Kale. A combinatorial, primal-dual approach to semidefinite programs. In *STOC*, pages 227–236, 2007.

[3] K. S. Azoury and M. K. Warmuth. Relative loss bounds for on-line density estimation with the exponential family of distributions. *Machine Learning*, 43(3):211–246, 2001.

[4] J. M. Bernardo and A. F. M. Smith. *Bayesian Theory*. Wiley, 1994.

[5] K. Bostroem and T. Felbinger. Lossless quantum data compression and variable-length coding. *Phys. Rev. A*, 65(3):032313, 2002.

[6] N. Cesa-Bianchi and G. Lugosi. *Prediction, Learning, and Games*. Cambridge University Press, New York, NY, USA, 2006.

[7] T. M. Cover and J. A. Thomas. *Elements of Information Theory*. John Wiley & Sons, 1991.

[8] R. Jain, Z. Ji, S. Upadhyay, and J. Watrous. QIP = PSPACE. In *Proceedings of the 42nd ACM Symposium on Theory of Computing, STOC*, pages 573–582, 2010.

[9] P. Kontkanen and P. Myllymäki. A fast normalized maximum likelihood algorithm for multinomial data. In *Proceedings of the Nineteenth International Joint Conference on Artificial Intelligence (IJCAI-05)*, pages 1613–1616, 2005.

[10] R. E. Krichevsky and V. K. Trofimov. The performance of universal encoding. *IEEE Transactions on Information Theory*, 27(2):199–207, Mar. 1981.

[11] J. T. Kwok and H. Zhao. Incremental eigen decomposition. In *IN PROC. ICANN*, pages 270–273, 2003.

[12] M. A. Nielsen and I. L. Chuang. *Quantum Computation and Quantum Information*. Cambridge University Press, 2000.

[13] B. Schumacher and M. D. Westmoreland. Indeterminate-length quantum coding. *Phys. Rev. A*, 64(4):042304, 2001.

[14] Y. M. Shtarkov. Universal sequential coding of single messages. *Problems of Information Transmission*, 23(3):3–17, 1987.

[15] M. Sion. On general minimax theorems. *Pacific Jouronal of Mathematics*, 8(1):171–176, 1958.

[16] E. Takimoto and M. Warmuth. The last-step minimax algorithm. In *Proceedings of the 13th Annual Conference on Computational Learning Theory*, pages 100–106, 2000.

[17] K. Tsuda, G. Rätsch, and M. K. Warmuth. Matrix exponentiated gradient updates for on-line learning and Bregman projections. *Journal of Machine Learning Research*, 6:995–1018, June 2005.

[18] M. K. Warmuth. When is there a free matrix lunch. In *Proc. of the 20th Annual Conference on Learning Theory (COLT '07)*. Springer-Verlag, June 2007. Open problem.

[19] M. K. Warmuth and D. Kuzmin. Online variance minimization. In *Proceedings of the 19th Annual Conference on Learning Theory (COLT '06)*, Pittsburg, June 2006. Springer-Verlag.

[20] M. K. Warmuth and D. Kuzmin. Bayesian generalized probability calculus for density matrices. *Journal of Machine Learning*, 78(1-2):63–101, January 2010.

